# Large-Scale Category Structure Aware Image Categorization

**Bin Zhao**
School of Computer Science
Carnegie Mellon University
binzhao@cs.cmu.edu

**Li Fei-Fei**
Computer Science Department
Stanford University
feifeili@cs.stanford.edu

**Eric P. Xing**
School of Computer Science
Carnegie Mellon University
epxing@cs.cmu.edu

## Abstract

Most previous research on image categorization has focused on medium-scale data sets, while large-scale image categorization with millions of images from thousands of categories remains a challenge. With the emergence of structured large-scale dataset such as the ImageNet, rich information about the conceptual relationships between images, such as a tree hierarchy among various image categories, become available. As human cognition of complex visual world benefits from underlying semantic relationships between object classes, we believe a machine learning system can and should leverage such information as well for better performance. In this paper, we employ such semantic relatedness among image categories for large-scale image categorization. Specifically, a category hierarchy is utilized to properly define loss function and select common set of features for related categories. An efficient optimization method based on proximal approximation and accelerated parallel gradient method is introduced. Experimental results on a subset of ImageNet containing 1.2 million images from 1000 categories demonstrate the effectiveness and promise of our proposed approach.

## 1 Introduction

Image categorization / object recognition has been one of the most important research problems in the computer vision community. While most previous research on image categorization has focused on medium-scale data sets, involving objects from dozens of categories, there is recently a growing consensus that it is necessary to build general purpose object recognizers that are able to recognize many more different classes of objects. (A human being has little problem recognizing tens of thousands of visual categories, even with very little "training" data.) The Caltech 101/256 [14, 18] is a pioneer benchmark data set on that front. LabelMe [31] provides 30k labeled and segmented images, covering around 200 image categories. Moreover, the newly released ImageNet [12] data set goes a big step further, in that it further increases the number of classes to over 15000, and has more than 1000 images for each class on average. Similarly, TinyImage [36] contains 80 million $32 \times 32$ low resolution images, with each image loosely labeled with one of 75,062 English nouns. Clearly, these are no longer artificial visual categorization problems created for machine learning, but instead more like a human-level cognition problem for real world object recognition with a much bigger set of objects. A natural way to formulate this problem is a *multi-way* or *multi-task* classification, but the seemingly standard formulation on such gigantic data set poses a completely new challenge both to computer vision and machine learning. Unfortunately, despite the well-known advantages and recent advancements of multi-way classification techniques [1, 19, 4] in machine learning, complexity concerns have driven most research on such super large-scale data set back to simple methods such as *nearest neighbor search* [6], *least square regression* [16] or learning thousands of binary classifiers [24].

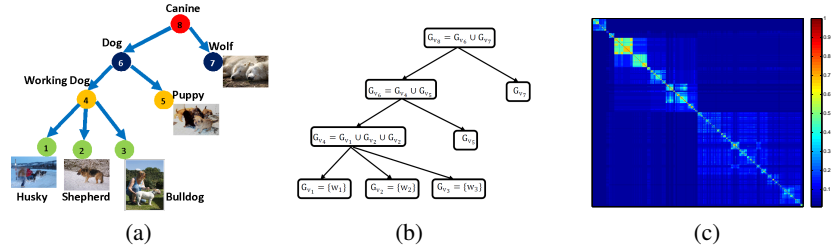

Figure 1: (a) Image category hierarchy in ImageNet; (b) Overlapping group structure; (3) Semantic relatedness measure between image categories.

The hierarchical semantic structure stemmed from the WordNet over image categories in the ImageNet data makes it distinctive from other existing large-scale dataset, and it reassembles how human cognitive system stores visual knowledge. Figure 1(a) shows an example such as a tree hierarchy, where leaf nodes are individual categories, and each internal node denotes the cluster of categories corresponding to the leaf nodes in the subtree rooted at the given node. As human cognition of complex visual world benefits from underlying semantic relationships between object classes, we believe a machine learning system can and should leverage such information as well for better performance. Specifically, we argue that instead of formulating the recognition task as a flat classification problem, where each category is treated equally and independently, a better strategy is to utilize the rich information residing in the concept hierarchy among image categories to train a system that couples all different recognition tasks over different categories. It should be noted that our proposed method is applicable to any tree structure for image category, such as the category structure learned to capture visual appearance similarities between image classes [32, 17, 13].

To the best of our knowledge, our attempt in this paper represents an initial foray to systematically utilizing information residing in concept hierarchy, for multi-way classification on super large-scale image data sets. More precisely, our approach utilizes the concept hierarchy in two aspects: loss function and feature selection. First, the loss function used in our formulation weighs differentially for different misclassification outcomes: misclassifying an image to a category that is close to its true identity should receive less penalty than misclassifying it to a totally unrelated one. Second, in an image classification problem with thousands of categories, it is not realistic to assume that all of the classes share the same set of relevant features. That is to say, a subset of highly related categories may share a common set of relevant features, whereas weakly related categories are less likely to be affected by the same features. Consequently, the image categorization problem is formulated as *augmented logistic regression with overlapping-group-lasso regularization*. The corresponding optimization problem involves a non-smooth convex objective function represented as summation over all training examples. To solve this optimization problem, we introduce the *Accelerated Parallel ProximaL gradiEnT (APPLET)* method, which tackles the non-smoothness of overlapping-group-lasso penalty via proximal gradient [20, 9], and the huge number of training samples by Map-Reduce parallel computing [10]. Therefore, the contributions made in this paper are: (1) We incorporate the semantic relationships between object classes, into an augmented multi-class logistic regression formulation, regularized by the overlapping-group-lasso penalty. The sheer size of the ImageNet data set that our formulation is designed to tackle singles out our work from previous attempts on multi-class classification, or transfer learning. (2) We propose a proximal gradient based method for solving the resulting non-smooth optimization problem, where the super large scale of the problem is tackled by map-reduce parallel computation.

The rest of this paper is organized as follows. Detailed explanation of the formulation is provided in Section 2. Section 3 introduces the *Accelerated Parallel ProximaL gradiEnT (APPLET)* method for solving the corresponding large-scale non-smooth optimization problem. Section 4 briefly reviews several related works. Section 5 demonstrates the effectiveness of the proposed algorithm using millions of training images from 1000 categories, followed by conclusions in Section 6.

## 2 Category Structure Aware Image Categorization

### 2.1 Motivation

ImageNet organizes the different classes of images in a densely populated semantic hierarchy. Specifically, image categories in ImageNet are interlinked by several types of relations, with the

"IS-A" relation being the most comprehensive and useful [11], resulting in a tree hierarchy over image categories. For example, the 'husky' category follows a path in the tree composed of 'working dog', 'dog', 'canine', etc. The distance between two nodes in the tree depicts the difference between the two corresponding image categories. Consequently, in the category hierarchy in ImageNet, each internal node near the bottom of the tree shows that the image categories of its subtree are highly correlated, whereas the internal node near the root represents relatively weaker correlations among the categories in its subtree.

The class hierarchy provides a measure of relatedness between image classes. Misclassifying an image to a category that is close to its true identity should receive less penalty than misclassifying it to a totally unrelated one. For example, although horses are not exactly ponies, we expect the loss for classifying a "pony" as a "horse" to be lower than classifying it as a "car". Instead of using 0-1 loss as in conventional image categorization, which treats image categories equally and independently, our approach utilizes a loss function that is aware of the category hierarchy.

Moreover, highly related image categories are more likely to share common visual patterns. For example, in Figure 1(a), *husky* and *shepherd* share similar object shape and texture. Consequently, recognition of these related categories are more likely to be affected by the same features. In this work, we regularize the sparsity pattern of weight vectors for related categories. This is equivalent to learning a low dimensional representation that is shared across multiple related categories.

## 2.2 Logistic Regression with Category Structure

Given $N$ training images, each represented as a $J$-dimensional input vector and belonging to one of the $K$ categories. Let $\mathbf{X}$ denote the $J \times N$ input matrix, where each column corresponds to an instance. Similarly, let $\mathbf{Y}$ denote the $N \times 1$ output vector, where each element corresponds to the label for an image. Multi-class logistic regression defines a weight vector $\mathbf{w}_k$ for each class $k \in \{1, \dots, K\}$ and classifies sample $\mathbf{z}$ by $y^* = \arg\max_{y \in \{1,\dots,k\}} P(y|\mathbf{x}, \mathbf{W})$, with the conditional likelihood computed as

$$P(y_i|\mathbf{x}_i, \mathbf{W}) = \frac{\exp(\mathbf{w}_{y_i}^T \mathbf{x}_i)}{\sum_k \exp(\mathbf{w}_k^T \mathbf{x}_i)} \tag{1}$$

The optimal weight vectors $\mathbf{W}^* = [\mathbf{w}_1^*, \dots, \mathbf{w}_K^*]$ are

$$\mathbf{W}^* = \arg\min_{\mathbf{W}} - \sum_{i=1}^N \log P(y_i|\mathbf{x}_i, \mathbf{W}) + \lambda\Omega(\mathbf{W}) \tag{2}$$

where $\Omega(\mathbf{W})$ is a regularization term defined on $\mathbf{W}$ and $\lambda$ is the regularization parameter.

### 2.2.1 Augmented Soft-Max Loss Function

Using the tree hierarchy on image categories, we could calculate a semantic relatedness (a.k.a. similarity) matrix $\mathbf{S} \in \mathbb{R}^{K \times K}$ over all categories, where $\mathbf{S}_{ij}$ measures the semantic relatedness of class $i$ and $j$. Using the semantic relatedness measure, the likelihood of $\mathbf{x}_i$ belonging to category $y_i$ could be modified as follows

$$\hat{P}(y_i|\mathbf{x}_i, \mathbf{W}) \propto \sum_{r=1}^K \mathbf{S}_{y_i,r} P(r|\mathbf{x}_i, \mathbf{W}) \propto \sum_{r=1}^K \mathbf{S}_{y_i,r} \frac{\exp(\mathbf{w}_r^T \mathbf{x}_i)}{\sum_k \exp(\mathbf{w}_k^T \mathbf{x}_i)} \propto \sum_{r=1}^K \mathbf{S}_{y_i,r} \exp(\mathbf{w}_r^T \mathbf{x}_i) \tag{3}$$

Since $\sum_{r=1}^K \hat{P}(r|\mathbf{x}_i, \mathbf{W}) = 1$, consequently,

$$\hat{P}(y_i|\mathbf{x}_i, \mathbf{W}) = \frac{\sum_{r=1}^K \mathbf{S}_{y_i,r} \exp(\mathbf{w}_r^T \mathbf{x}_i)}{\sum_{r=1}^K \sum_{k=1}^K \mathbf{S}_{k,r} \exp(\mathbf{w}_r^T \mathbf{x}_i)} \tag{4}$$

For the special case where the semantic relatedness matrix $\mathbf{S}$ is an identity matrix, meaning each class is only related to itself, Eq. (4) simplifies to Eq. (1). Using this modified softmax loss function, the image categorization problem could be formulated as

$$\min_{\mathbf{W}} \sum_{i=1}^N \left[ \log \left( \sum_r \sum_k \mathbf{S}_{k,r} \exp(\mathbf{w}_r^T \mathbf{x}_i) \right) - \log \left( \sum_r \mathbf{S}_{y_i,r} \exp(\mathbf{w}_r^T \mathbf{x}_i) \right) \right] + \lambda\Omega(\mathbf{W}) \tag{5}$$

### 2.2.2 Semantic Relatedness Matrix

To compute semantic relatedness matrix $\mathbf{S}$ in the above formulation, we first define a metric measuring the semantic distance between image categories. A simple way to compute semantic distance in a structure such as the one provided by ImageNet is to utilize the paths connecting the two corresponding nodes to the root node. Following [7] we define the semantic distance $\mathbf{D}_{ij}$ between class $i$ and class $j$ as the number of nodes shared by their two parent branches, divided by the length of the longest of the two branches

$$\mathbf{D}_{ij} = \frac{\mathrm{intersect}(\mathrm{path}(i), \mathrm{path}(j))}{\max(\mathrm{length}(\mathrm{path}(i)), \mathrm{length}(\mathrm{path}(j)))} \tag{6}$$

where $\mathrm{path}(i)$ is the path from the root node to node $i$ and $\mathrm{intersect}(p_1, p_2)$ counts the number of nodes shared by two paths $p_1$ and $p_2$. We construct the semantic relatedness matrix $\mathbf{S} = \exp(-\kappa(1 - \mathbf{D}))$, where $\kappa$ is a constant controlling the decay factor of semantic relatedness with respect to semantic distance. Figure 1(c) shows the semantic relatedness matrix computed with $\kappa = 5$.

### 2.3 Tree-Guided Sparse Feature Coding

In ImageNet, image categories are grouped at multiple granularity as a tree hierarchy. As illustrated in Section 2.1, the image categories in each internal node are likely to be influenced by a common set of features. In order to achieve this type of structured sparsity at multiple levels of the hierarchy, we utilize an overlapping-group-lasso penalty recently proposed in [21] for genetic association mapping problem, where the goal is to identify a small number of SNPs (inputs) out of millions of SNPs that influence phenotypes (outputs) such as gene expression measurements.

Specifically, given the tree hierarchy $\mathcal{T} = (\mathcal{V}, \mathcal{E})$ over image categories, each node $v \in \mathcal{V}$ of tree $\mathcal{T}$ is associated with group $G_v$, composed of all leaf nodes in the subtree rooted at $v$, as illustrated in Figure 1(b). Clearly, each group $G_v$ is a subset of the power set of $\{1, \ldots, K\}$. Given these groups $\mathcal{G} = \{G_v\}_{v \in \mathcal{V}}$ of categories, we define the following overlapping-group-lasso penalty [21]:

$$\Omega(\mathbf{W}) = \sum_j \sum_{v \in \mathcal{V}} \gamma_v ||\mathbf{w}_{jG_v}||_2 \tag{7}$$

where $\mathbf{w}_{jG_v}$ is the weight coefficients $\{w_{jk}, k \in G_v\}$ for input $j \in \{1, \ldots, J\}$ associated with categories in $G_v$, and each group $G_v$ is associated with weight $\gamma_v$ that reflects the strength of correlation within the group. It should be noted that we do not require groups in $\mathcal{G}$ to be mutually exclusive, and consequently, each leaf node would belong to multiple groups at various granularity.

Inserting the above overlapping-group-lasso penalty into (5), we formulate the category structure aware image categorization as follows:

$$\min_{\mathbf{W}} \sum_{i=1}^N \left[ \log\left( \sum_r \sum_k \mathbf{S}_{k,r} \exp(\mathbf{w}_r^T \mathbf{x}_i) \right) - \log\left( \sum_r \mathbf{S}_{y_i,r} \exp(\mathbf{w}_r^T \mathbf{x}_i) \right) \right] + \lambda \sum_j \sum_{v \in \mathcal{V}} \gamma_v ||\mathbf{w}_{G_v}^j||_2 \tag{8}$$

## 3 Accelerated Parallel ProximaL gradiEnT (APPLET) Method

The challenge in solving problem (8) lies in two facts: the non-separability of $\mathbf{W}$ in the non-smooth overlapping-group-lasso penalty $\Omega(\mathbf{W})$, and the huge number $N$ of training samples. Conventionally, to handle the non-smoothness of $\Omega(\mathbf{W})$, we could reformulate the problem as either *second order cone programming (SOCP)* or *quadratic programming (QP)* [35]. However, the state-of-the-art approach for solving *SOCP* and *QP* based on *interior point method* requires solving a Newton system to find search direction, and is computationally very expensive even for moderate-sized problems. Moreover, due to the huge number of samples in the training set, off-the-shelf optimization solvers are too slow to be used.

In this work, we adopt a proximal-gradient method to handle the non-smoothness of $\Omega(\mathbf{W})$. Specifically, we first reformulate the overlapping-group-lasso penalty $\Omega(\mathbf{W})$ into a $\max$ problem over auxiliary variables using dual norm, and then introduce its smooth lower bound [20, 9]. Instead of optimizing the original non-smooth penalty, we run the *accelerated gradient descent* method [27] under a Map-Reduce framework [10] to optimize the smooth lower bound. The proposed approach enjoys a fast convergence rate and low per-iteration complexity.

## 3.1 Reformulate the Penalty

For referring convenience, we number the elements in the set $\mathcal{G} = \{G_v\}_{v \in \mathcal{V}}$ as $\mathcal{G} = \{\mathbf{g}_1, \ldots, \mathbf{g}_{|\mathcal{G}|}\}$ according to an arbitrary order, where $|\mathcal{G}|$ denotes the total number of elements in $\mathcal{G}$. For each input $j$ and group $\mathbf{g}_i$ associated with $\mathbf{w}_{j\mathbf{g}_i}$, we introduce a vector of auxiliary variables $\boldsymbol{\alpha}_{j\mathbf{g}_i} \in \mathbb{R}^{|\mathbf{g}_i|}$. Since the dual norm of $L_2$ norm is also an $L_2$ norm, we can reformulate $||\mathbf{w}_{j\mathbf{g}_i}||_2$ as $||\mathbf{w}_{j\mathbf{g}_i}||_2 = \max_{||\boldsymbol{\alpha}_{j\mathbf{g}_i}||_2 \leq 1} \boldsymbol{\alpha}_{j\mathbf{g}_i}^T \mathbf{w}_{j\mathbf{g}_i}$. Moreover, define the following $\sum_{\mathbf{g} \in \mathcal{G}} |\mathbf{g}| \times J$ matrix

$$\mathbf{A} = \begin{pmatrix} \boldsymbol{\alpha}_{1\mathbf{g}_1} & \cdots & \boldsymbol{\alpha}_{J\mathbf{g}_1} \\ \vdots & \ddots & \vdots \\ \boldsymbol{\alpha}_{1\mathbf{g}_{|\mathcal{G}|}} & \cdots & \boldsymbol{\alpha}_{J\mathbf{g}_{|\mathcal{G}|}} \end{pmatrix} \tag{9}$$

in domain $\mathcal{O} = \{\mathbf{A}| \, ||\boldsymbol{\alpha}_{j\mathbf{g}_i}||_2 \leq 1, \forall j \in \{1, \ldots, J\}, \mathbf{g}_i \in \mathcal{G}\}$. Following [9], the overlapping-group-lasso penalty in (8) can be equivalently reformulated as

$$\Omega(\mathbf{W}) = \sum_j \sum_i \gamma_i \max_{||\boldsymbol{\alpha}_{j\mathbf{g}_i}||_2 \leq 1} \boldsymbol{\alpha}_{j\mathbf{g}_i}^T \mathbf{w}_{j\mathbf{g}_i} = \max_{A \in \mathcal{O}} \langle \mathbf{C}\mathbf{W}^T, \mathbf{A} \rangle \tag{10}$$

where $i = 1, \ldots, |\mathcal{G}|$, $j = 1, \ldots, J$, $\mathbf{C} \in \mathbb{R}^{\sum_{\mathbf{g} \in \mathcal{G}} |\mathbf{g}| \times K}$, and $\langle \mathbf{U}, \mathbf{V} \rangle = \text{Tr}(\mathbf{U}^T \mathbf{V})$ is the inner product of two matrices. Moreover, the matrix $\mathbf{C}$ is defined with rows indexed by $(s, \mathbf{g}_i)$ such that $s \in \mathbf{g}_i$ and $i \in \{1, \ldots, |\mathcal{G}|\}$, columns indexed by $k \in \{1, \ldots, K\}$, and the value of the element at row $(s, \mathbf{g}_i)$ and column $k$ set to $\mathbf{C}_{(s,\mathbf{g}_i),k} = \gamma_i$ if $s = k$ and $0$ otherwise.

After the above reformulation, (10) is still a non-smooth function of $\mathbf{W}$, and this makes the optimization challenging. To tackle this problem, we introduce an auxiliary function [20, 9] to construct a smooth approximation of (10). Specifically, our smooth approximation function is defined as:

$$f_\mu(\mathbf{W}) = \max_{\mathbf{A} \in \mathcal{O}} \langle \mathbf{C}\mathbf{W}^T, \mathbf{A} \rangle - \mu d(\mathbf{A}) \tag{11}$$

where $\mu$ is the positive smoothness parameter and $d(\mathbf{A})$ is an arbitrary smooth strongly-convex function defined on $\mathcal{O}$. The original penalty term can be viewed as $f_\mu(\mathbf{W})$ with $\mu = 0$. Since our algorithm will utilize the optimal solution $\mathbf{W}^*$ to (11), we choose $d(\mathbf{A}) = \frac{1}{2}||\mathbf{A}||_F^2$ so that we can obtain the closed form solution for $\mathbf{A}^*$. Clearly, $f_\mu(\mathbf{W})$ is a lower bound of $f_0(\mathbf{W})$, with the gap computed as $D = \max_{\mathbf{A} \in \mathcal{O}} d(\mathbf{A}) = \max_{\mathbf{A} \in \mathcal{O}} \frac{1}{2}||\mathbf{A}||_F^2 = \frac{1}{2}J|\mathcal{G}|$.

**Theorem 1** *For any $\mu > 0$, $f_\mu(\mathbf{W})$ is a convex and continuously differentiable function in $\mathbf{W}$, and the gradient of $f_\mu(\mathbf{W})$ can be computed as $\nabla f_\mu(\mathbf{W}) = \mathbf{A}^{*T}\mathbf{C}$, where $A^*$ is the optimal solution to (11).*

According to Theorem 1, $f_\mu(\mathbf{W})$ is a smooth function for any $\mu > 0$, with a simple form of gradient and can be viewed as a smooth approximation of $f_0(\mathbf{W})$ with the maximum gap of $\mu D$. Finally, the optimal solution $\mathbf{A}^*$ of (11) is composed of $\boldsymbol{\alpha}_{j\mathbf{g}_i}^* = S(\frac{\gamma_i \mathbf{w}_{j\mathbf{g}_i}}{\mu})$, where $S$ is the shrinkage operator defined as follows:

$$S(\mathbf{u}) = \begin{cases} \frac{\mathbf{u}}{||\mathbf{u}||_2}, & ||\mathbf{u}||_2 > 1 \\ \mathbf{u}, & ||\mathbf{u}||_2 \leq 1 \end{cases} \tag{12}$$

## 3.2 Accelerated Parallel Gradient Method

Given the smooth approximation of $\Omega(\mathbf{W})$ in (11) and the corresponding gradient presented in Theorem 1, we could apply *gradient descent* method to solve the problem. Specifically, we replace the overlapping-group-lasso penalty in (8) with its smooth approximation $f_\mu(\mathbf{W})$ to obtain the following optimization problem

$$\min_{\mathbf{W}} \tilde{f}(\mathbf{W}) = g(\mathbf{W}) + \lambda f_\mu(\mathbf{W}) \tag{13}$$

where $g(\mathbf{W}) = \sum_{i=1}^N \left[ \log\left(\sum_r \sum_k \mathbf{S}_{k,r} \exp(\mathbf{w}_r^T \mathbf{x}_i)\right) - \log\left(\sum_r \mathbf{S}_{y_i,r} \exp(\mathbf{w}_r^T \mathbf{x}_i)\right) \right]$ is the augmented logistic regression loss function. The gradient of $g(\mathbf{W})$ w.r.t. $\mathbf{w}_k$ could be calculated as follows

$$\frac{\partial g(\mathbf{W})}{\partial \mathbf{w}_k} = \sum_{i=1}^N \mathbf{x}_i \left[ \frac{\sum_q \mathbf{S}_{k,q} \exp(\mathbf{w}_k^T \mathbf{x}_i)}{\sum_r \sum_q \mathbf{S}_{r,q} \exp(\mathbf{w}_r^T \mathbf{x}_i)} - \frac{\mathbf{S}_{y_i,k} \exp(\mathbf{w}_k^T \mathbf{x}_i)}{\sum_r \mathbf{S}_{y_i,r} \exp(\mathbf{w}_r^T \mathbf{x}_i)} \right] \tag{14}$$

Therefore, the gradient of $g(\mathbf{W})$ w.r.t. to $\mathbf{W}$ could be computed as $\nabla g(\mathbf{W}) = [\frac{\partial g(\mathbf{W})}{\partial \mathbf{w}_1}, \ldots, \frac{\partial g(\mathbf{W})}{\partial \mathbf{w}_K}]$. According to Theorem 1, the gradient of $\tilde{f}(\mathbf{W})$ is given by

$$\nabla \tilde{f}(\mathbf{W}) = \nabla g(\mathbf{W}) + \lambda \mathbf{A}^{*T} \mathbf{C} \tag{15}$$

Although $\tilde{f}(\mathbf{W})$ is a smooth function of $\mathbf{W}$, it is represented as a summation over all training samples. Consequently, $\nabla \tilde{f}(\mathbf{W})$ could only be computed by summing over all $N$ training samples. Due to the huge number of samples in the training set, we adopt a Map-Reduce parallel framework [10] to compute $\nabla g(\mathbf{W})$ as shown in Eq.(14). While standard gradient schemes have a slow convergence rate, they can often be accelerated. This stems from the pioneering work of Nesterov in [27], which is a deterministic algorithm for smooth optimization. In this paper, we adopt this accelerated gradient method , and the whole algorithm is shown in Algorithm 1.

---

**Algorithm 1** Accelerated Parallel ProximaL gradiEnT method (APPLET)

---

    **Input**: $\mathbf{X}$, $\mathbf{Y}$, $\mathbf{C}$, desired accuracy $\epsilon$, step parameters $\{\eta_t\}$
    **Initialization**: $\mathbf{B}_0 = \mathbf{0}$
    **for** $t = 1, 2, \ldots$, until convergence **do**
        **Map-step**: Distribute data to $M$ cores $\{\mathcal{X}_1, \ldots, \mathcal{X}_M\}$, compute in parallel $\nabla g_m(\mathbf{B}_{t-1})$ for $\mathcal{X}_m$
        **Reduce-step**:
        (1) $\nabla \tilde{f}(\mathbf{B}_{t-1}) = \sum_{m=1}^{M} \nabla g_m(\mathbf{B}_{t-1}) + \lambda \mathbf{A}^{*T} \mathbf{C}$
        (2) $\mathbf{W}_t = \mathbf{B}_{t-1} - \eta_t \nabla \tilde{f}(\mathbf{B}_{t-1})$
        (3) $\mathbf{B}_t = \mathbf{W}_t + \frac{t-1}{t+2}(\mathbf{W}_t - \mathbf{W}_{t-1})$
    **end for**
    **Output**: $\hat{\mathbf{W}} = \mathbf{W}_t$

---

## 4 Related Works

Various attempts in sharing information across related image categories have been explored. Early approaches stem from the neural networks, where the hidden layers are shared across different classes [8, 23]. Recent approaches transfer information across classes by regularizing the parameters of the classifiers across classes [37, 28, 15, 33, 34, 2, 26, 30]. Common to all these approaches is that experiments are always performed with relatively few classes [16]. It is unclear how these approaches would perform on super large-scale data sets containing thousands of image categories. Some of these approaches would encounter severe computational bottleneck when scaling up to thousands of classes [16].

Another line of research is the ImageNet Large Scale Visual Recognition Challenge 2010 (ILSVRC10) [3], where best performing approaches use techniques such as spatial pyramid matching [22], locality-constrained linear coding [38], the Fisher vector [29], and linear SVM trained using stochastic gradient descent. Success has been witnessed in ILSVRC10 even with simple machine learning techniques. However, none of these approaches utilize the semantic relationships defined among image categories in ImageNet, which we argue is a crucial source of information for further improvement in such super large scale classification problem.

## 5 Experiments

In this section, we test the performance of *APPLET* on a subset of ImageNet used in ILSVRC10, containing 1.2 million images from 1000 categories, divided into distinct portions for training, validation and test. The number of images for each category ranges from 668 to 3047. We use the provided validation set for parameter selection and the final results are obtained on the test set.

Before presenting the classification results, we'd like to make clear that the goal and contributions of this work is different from the aforementioned approaches proposed in ILSVRC10. Those approaches were designed to enter a performance competition, where heavy feature engineering and post processing (such as ad hoc voting for multiple algorithms) were used to achieve high accuracy. Our work, on the other hand, looks at this problem from a different angle, focusing on principled

methodology that explores the benefit of utilizing class structure in image categorization and proposing a model and related optimization technique to properly incorporate such information. We did not use the full scope of all the features, and post processing schemes to boost our classification results as the ILSVRC10 competition teams did. Therefore we argue that the results of our work is not directly comparable with the ILSVRC10 competitions.

## 5.1 Image Features

Each image is resized to have a max side length of 300 pixels. SIFT [25] descriptors are computed on $20 \times 20$ overlapping patches with a spacing of 10 pixels. Images are further downsized to $\frac{1}{2}$ of the side length and then $\frac{1}{4}$ of the side length, and more descriptors are computed. We then perform k-means clustering on a random subset of 10 million SIFT descriptors to form a visual vocabulary of 1000 visual words. Using this learned vocabulary, we employ *Locality-constrained Linear Coding (LLC)* [38], which has shown state-of-the-art performance on several benchmark data sets, to construct a vector representation for each image. Finally, a single feature vector is computed for each image using max pooling on a spatial pyramid [22]. The pooled features from various locations and scales are then concatenated to form a spatial pyramid representation of the image. Consequently, each image is represented as a vector in a 21,000 dimensional space.

## 5.2 Evaluation Criteria

We adopt the same performance measures used in ILSVRC10. Specifically, for every image, each tested algorithm will produce a list of 5 object categories in the descending order of confidence. Performance is measured using the top-$n$ error rate, $n = 1, \ldots, 5$ in our case, and two error measures are reported. The first is a **flat error** which equals 1 if the true class is not within the $n$ most confident predictions, and 0 otherwise. The second is a **hierarchical error**, reporting the minimum height of the lowest common ancestors between true and predicted classes. For each of the above two criteria, the overall error score for an algorithm is the average error over all test images.

Table 1: Classification results (both flat and hierarchical errors) of various algorithms.

| Algorithm | Flat Error | | | | | Hierarchical Error | | | | |
|---|---|---|---|---|---|---|---|---|---|---|
| | Top 1 | Top 2 | Top 3 | Top 4 | Top 5 | Top 1 | Top 2 | Top 3 | Top 4 | Top 5 |
| LR | 0.797 | 0.726 | 0.678 | 0.639 | 0.607 | 8.727 | 6.974 | 5.997 | 5.355 | 4.854 |
| ALR | 0.796 | 0.723 | 0.668 | 0.624 | 0.587 | 8.259 | 6.234 | 5.061 | 4.269 | 3.659 |
| GroupLR | 0.786 | 0.699 | 0.642 | 0.600 | 0.568 | 7.620 | 5.460 | 4.322 | 3.624 | 3.156 |
| APPLET | 0.779 | 0.698 | 0.634 | 0.589 | 0.565 | 7.208 | 4.985 | 3.798 | 3.166 | 3.012 |

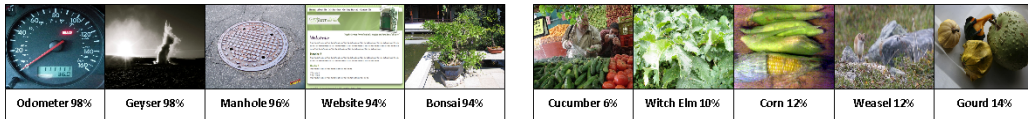

Figure 2: Left: image classes with highest accuracy. Right: image classes with lowest accuracy.

## 5.3 Comparisons & Classification Results

We have conducted comprehensive performance evaluations by testing our method under different circumstances. Specifically, to better understand the effect of augmenting logistic regression with semantic relatedness and use of overlapping-group-lasso penalty to enforce group level feature selection, we study the model adding only augmented logistic regression loss and adding only overlapping-group-lasso penalty separately, and compare with the *APPLET* method. We use the conventional $L_2$ regularized logistic regression [5] as baseline. The algorithms that we evaluated are listed below: (1)$L_2$ regularized logistic regression (LR) [5]; (2) Augmented logistic regression with $L_2$ regularization (ALR); (3) Logistic regression with overlapping-group-lasso regularization (GroupLR); (4) Augmented logistic regression with overlapping-group-lasso regularization (APPLET).

Table 1 presents the classification results of various algorithms. According to the classification results, we could clearly see the advantage of *APPLET* over conventional logistic regression, especially on the top-5 error rate. Specifically, comparing the top-5 error rate, *APPLET* outperforms *LR* by a margin of $0.04$ on flat loss, and a margin of $1.84$ on hierarchical loss. It should be noted

that hierarchical error is measured by the height of the lowest common ancestor in the hierarchy, and moving up a level can more than double the number of descendants. Table 1 also compares the performance of *ALR* with *LR*. Specifically, *ALR* outperforms *LR* slightly when using the top-1 prediction results. However, on top-5 prediction results, *ALR* performs clearly better than *LR*. Similar phenomenon is observed when comparing the classification results of *GroupLR* with *LR*. Moreover, Figure 2 shows the image categories with highest and lowest classification accuracy.

One key reason for introducing the augmented loss function is to ensure that predicted image class falls not too far from its true class on the semantic hierarchy. Results in Table 2 demonstrate that even though *APPLET* cannot guarantee to make the correct prediction on each image, it produces labels that are closer to the true one than *LR*, which generates labels far from correct ones.

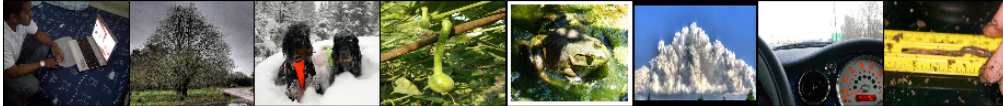

| True class | laptop | linden | gordon setter | gourd | bullfrog | volcano | odometer | earthworm |
|---|---|---|---|---|---|---|---|---|
| APPLET | laptop(0) | live oak(3) | Irish setter(2) | acorn(2) | woodfrog(2) | volcano(0) | odometer(0) | earthworm(0) |
| LR | laptop(0) | log wood(3) | alp(11) | olive(2) | water snake(9) | geyser(4) | odometer(0) | slug(8) |

Table 2: Example prediction results of *APPLET* and *LR*. Numbers indicate the **hierarchical error** of the misclassification, defined in Section 5.2.

As shown in Table 1, a systematic reduction in classification error using *APPLET* shows that acknowledging semantic relationships between image classes enables the system to discriminate at more informative semantic levels. Moreover, results in Table 2 demonstrate that classification results of *APPLET* can be significantly more informative, as labeling a "bullfrog" as "woodfrog" gives a more useful answer than "water snake", as it is still correct at the "frog" level.

## 5.4   Effects of $\lambda$ and $\kappa$ on the Performance of *APPLET*

We present in Figure 3 how categorization performance scales with $\lambda$ and $\kappa$. According to Figure 3, *APPLET* achieves lowest categorization error around $\lambda = 0.01$. Moreover, the error rate increases

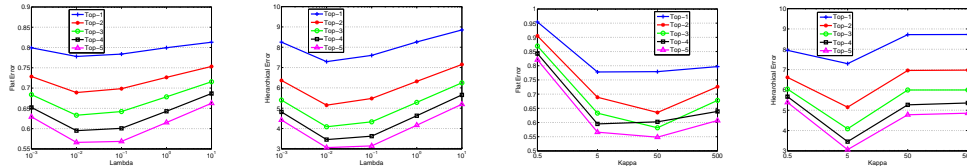

Figure 3: Classification results (flat error and hierarchical error) of *APPLET* with various $\lambda$ and $\kappa$.

when $\lambda$ is larger than 0.1, when excessive regularization hampers the algorithm from differentiating semantically related categories. Similarly, *APPLET* achieves best performance with $\kappa = 5$. When $\kappa$ is too small, a large number of categories are mixed together, resulting in a much higher flat loss. On the other hand, when $\kappa \geq 50$, the semantic relatedness matrix is close to diagonal, resulting in treating all categories independently, and categorization performance becomes similar as *LR*.

## 6   Conclusions

In this paper, we argue the positive effect of incorporating category hierarchy information in super large scale image categorization. The sheer size of the problem considered here singles out our work from any previous works on multi-way classification or transfer learning. Empirical study using 1.2 million training images from 1000 categories demonstrates the effectiveness and promise of our proposed approach.

**Acknowledgments**

E. P. Xing is supported by NSF IIS-0713379, DBI-0546594, Career Award, ONR N000140910758, DARPA NBCH1080007 and Alfred P. Sloan Foundation. L. Fei-Fei is partially supported by an NSF CAREER grant (IIS-0845230) and an ONR MURI grant.

# References

[1] B. Bakker and T. Heskes. Task clustering and gating for bayesian multitask learning. *JMLR*, 4:83–99, 2003.

[2] E. Bart and S. Ullman. Cross-generalization: learning novel classes from a single example by feature replacement. In *CVPR*, 2005.

[3] A. Berg, J. Deng, and L. Fei-Fei. Large scale visual recognition challenge 2010. http://www.image-net.org/challenges/LSVRC/2010/, 2010.

[4] A. Binder, K.-R. Mller, and M. Kawanabe. On taxonomies for multi-class image categorization. *IJCV*, pages 1–21, 2011.

[5] C. Bishop. *Pattern Recognition and Machine Learning*. Springer-Verlag New York, Inc., 2006.

[6] O. Boiman, E. Shechtman, and M. Irani. In defense of nearest-neighbor based image classification. In *CVPR*, 2008.

[7] A. Budanitsky and G. Hirst. Evaluating wordnet-based measures of lexical semantic relatedness. *Comput. Linguist.*, 32:13–47, March 2006.

[8] R. Caruana. Multitask learning. *Machine Learning*, 28:41–75, 1997.

[9] X. Chen, Q. Lin, S. Kim, J. Carbonell, and E. P. Xing. Smoothing proximal gradient method for general structured sparse learning. In *UAI*, 2011.

[10] C. Chu, S. Kim, Y. Lin, Y. Yu, G., A. Ng, and K. Olukotun. Map-reduce for machine learning on multicore. In *NIPS*. 2007.

[11] J. Deng, A. Berg, K. Li, and L. Fei-Fei. What does classifying more than 10,000 image categories tell us? In *ECCV*, 2010.

[12] J. Deng, W. Dong, R. Socher, L.-J. Li, K. Li, and L. Fei-Fei. ImageNet: A Large-Scale Hierarchical Image Database. In *CVPR*, 2009.

[13] J. Deng, S. Satheesh, A. Berg, and L. Fei-Fei. Fast and balanced: Efficient label tree learning for large scale object recognition. In *NIPS*, 2011.

[14] L. Fei-Fei, R. Fergus, and P. Perona. Learning generative visual models from few training examples: an incremental bayesian approach tested on 101 object categories. In *CVPR Workshop on Generative-Model Based Vision*, 2004.

[15] L. Fei-Fei, R. Fergus, and P. Perona. One-shot learning of object categories. *PAMI*, 28:594–611, 2006.

[16] R. Fergus, H. Bernal, Y. Weiss, and A. Torralba. Semantic label sharing for learning with many categories. In *ECCV*, ECCV'10, 2010.

[17] T. Gao and D. Koller. Discriminative learning of relaxed hierarchy for large-scale visual recognition. In *ICCV*, 2011.

[18] G. Griffin, A. Holub, and P. Perona. Caltech-256 object category dataset. *Technical Report 7694, California Institute of Technology*, 2007.

[19] L. Jacob, F. Bach, and J.-P. Vert. Clustered multi-task learning: A convex formulation. In *NIPS*, 2008.

[20] R. Jenatton, J. Mairal, G. Obozinski, and F. Bach. Proximal methods for sparse hierarchical dictionary learning. In *ICML*, 2010.

[21] S. Kim and E. Xing. Tree-guided group lasso for multi-task regression with structured sparsity. In *ICML*, 2010.

[22] S. Lazebnik, C. Schmid, and J. Ponce. Beyond bags of features: Spatial pyramid matching for recognizing natural scene categories. In *CVPR*, 2006.

[23] Y. LeCun, L. Bottou, Y. Bengio, and P. Haffner. Gradient-based learning applied to document recognition. *Proc. IEEE*, 86:2278–2324, 1998.

[24] Y. Lin, F. Lv, S. Zhu, M. Yang, T. Cour, K. Yu, L. Cao, and T. Huang. Large-scale image classification: fast feature extraction and svm training. In *CVPR*, 2011.

[25] D. Lowe. Distinctive image features from scale-invariant keypoints. *IJCV*, 60:91–110, 2004.

[26] E. Miller, N. Matsakis, and P. Viola. Learning from one example through shared densities on transforms. In *CVPR*, 2000.

[27] Y. Nesterov. A method for unconstrained convex minimization problem with the rate of convergence $o(\frac{1}{k^2})$. *Doklady AN SSSR (translated as Soviet. Math. Docl.)*, 269:543–547, 1983.

[28] A. Opelt, A. Pinz, and A. Zisserman. Incremental learning of object detectors using a visual shape alphabet. In *CVPR*, 2006.

[29] F. Perronnin, J. Sanchez, and T. Mensink. Improving the fisher kernel for large-scale image classification. In *ECCV*, 2010.

[30] A. Quattoni, M. Collins, and T. Darrell. Transfer learning for image classification with sparse prototype representations. In *CVPR*, 2008.

[31] B. Russell, A. Torralba, K. Murphy, and W. Freeman. Labelme: A database and web-based tool for image annotation. *IJCV*, 77:157–173, 2008.

[32] R. Salakhutdinov, A. Torralba, and Josh Tenenbaum. Learning to share visual appearance for multiclass object detection. In *CVPR*, 2011.

[33] E. Sudderth, A. Torralba, W. Freeman, and A. Willsky. Learning hierarchical models of scenes, objects, and parts. In *CVPR*, 2005.

[34] J. Tenenbaum and W. Freeman. Separating style and content with bilinear models. *Neural Computation*, 12:1247–1283, 2000.

[35] R. Tibshirani, M. Saunders, S. Rosset, J. Zhu, and K. Knight. Sparsity and smoothness via the fused lasso. *Journal of the Royal Statistical Society Series B*, pages 91–108, 2005.

[36] A. Torralba, R. Fergus, and W. Freeman. 80 million tiny images: A large data set for nonparametric object and scene recognition. *PAMI*, 30:1958–1970, 2008.

[37] A. Torralba, K. Murphy, and W. Freeman. Sharing features: efficient boosting procedures for multiclass object detection. In *CVPR*, 2004.

[38] J. Wang, J. Yang, K. Yu, F. Lv, T. Huang, and Y. Gong. Locality-constrained linear coding for image classification. In *CVPR*, 2010.

